# Co-Training for Domain Adaptation

**Minmin Chen, Kilian Q. Weinberger**
Department of Computer Science and Engineering
Washington University in St. Louis
St. Louis, MO 63130
mc15,kilian@wustl.edu

**John C. Blitzer**
Google Research
1600 Amphitheatre Parkway
Mountain View, CA 94043
blitzer@google.com

## Abstract

Domain adaptation algorithms seek to generalize a model trained in a *source* domain to a new *target* domain. In many practical cases, the source and target distributions can differ substantially, and in some cases crucial target features may not have support in the source domain. In this paper we introduce an algorithm that bridges the gap between source and target domains by slowly adding to the training set both the target features and instances in which the current algorithm is the most confident. Our algorithm is a variant of co-training [7], and we name it CODA (Co-training for domain adaptation). Unlike the original co-training work, we do not assume a particular feature split. Instead, for each iteration of co-training, we formulate a single optimization problem which simultaneously learns a target predictor, a split of the feature space into views, and a subset of source and target features to include in the predictor. CODA significantly out-performs the state-of-the-art on the 12-domain benchmark data set of Blitzer et al. [4]. Indeed, over a wide range (65 of 84 comparisons) of target supervision CODA achieves the best performance.

## 1   Introduction

Domain adaptation addresses the problem of generalizing from a *source* distribution for which we have ample labeled training data to a *target* distribution for which we have little or no labels [3, 14, 28]. Domain adaptation is of practical importance in many areas of applied machine learning, ranging from computational biology [17] to natural language processing [11, 19] to computer vision [23].

In this work, we focus primarily on domain adaptation problems that are characterized by missing features. This is often the case in natural language processing, where different genres often use very different vocabulary to describe similar concepts. For example, in our experiments we use the sentiment data of Blitzer et al. [4], where *a breeze to use* is a way to express positive sentiment about kitchen appliances, but not about books. In this situation, most domain adaptation algorithms seek to eliminate the difference between source and target distributions, either by re-weighting source instances [14, 18] or learning a new feature representation [6, 28].

We present an algorithm which differs from both of these approaches. Our method seeks to slowly adapt its training set from the source to the target domain, using ideas from co-training. We accomplish this in two ways: First, we train on our own output in rounds, where at each round, we include in our training data the target instances we are most confident of. Second, we select a subset of shared source and target features based on their compatibility. Different from most previous work on selecting features for domain adaptation, the compatibility is measured across the training set and the unlabeled set, instead of across the two domains. As more target instances are added to the training set, target specific features become compatible across the two sets, therefore are included in the predictor. Finally, we exploit the pseudo multiview co-training algorithm of Chen *et al.* [10]

to exploit the unlabeled data efficiently. These three intuitive ideas can be combined in a single optimization problem. We name our algorithm CODA (Co-Training for Domain Adaptation).

By allowing us to slowly change our training data from source to target, CODA has an advantage over representation-learning algorithms [6, 28], since they must decide a priori what the best representation is. In contrast, each iteration of CODA can choose exactly those few target features which can be related to the current (source and pseudo-labeled target) training set. We find that in the sentiment prediction data set of Blitzer et al. [4] CODA improves the state-of-the-art cross widely varying amounts of target labeled data in 65 out of 84 settings.

## 2 Notation and Setting

We assume our data originates from two domains, Source (S) and Target (T). The source data is fully labeled $D_S = \{(\mathbf{x}_1, y_1), \ldots, (\mathbf{x}_{n_s}, y_{n_s})\} \subset \mathcal{R}^d \times \mathcal{Y}$ and sampled from some distribution $P_S(X, Y)$. The target data is sampled from $P_T(X, Y)$ and is divided into labeled $D_T^l = \{(\mathbf{x}_1, y_1), \ldots, (\mathbf{x}_{n_t}, y_{n_t})\} \subset \mathcal{R}^d \times \mathcal{Y}$ and unlabeled $D_T^u = \{(\mathbf{x}_1, ?), \ldots (\mathbf{x}_{m_t}, ?)\} \subset \mathcal{R}^d \times \mathcal{Y}$ parts, where in the latter the labels are unknown during training time. Both domains are of equal dimensionality $d$. Our goal is to learn a classifier $h \in \mathcal{H}$ to accurately predict the labels on the unlabeled portion of $D_T$, but also to extend to out-of-sample test points, such that for any $(\mathbf{x}, y)$ sampled from $P_T$, we have $h(\mathbf{x}) = y$ with high probability. For simplicity we assume that $\mathcal{Y} = \{+1, -1\}$, although our method can easily be adapted to multi-class or regression settings.

We assume the existence of a base classifier, which determines the set $\mathcal{H}$. Throughout this paper we simply use logistic regression, *i.e.* our classifier is parameterized by a weight-vector $\mathbf{w} \in \mathcal{R}^d$ and defined as $h_\mathbf{w}(\mathbf{x}) = (1 + e^{-\mathbf{w}^\top \mathbf{x}})^{-1}$. The weights $\mathbf{w}$ are set to minimize the loss function

$$\ell(\mathbf{w}; D) = -\frac{1}{|D|} \sum_{(\mathbf{x}, y) \in D} \log(1 + \exp(-y\mathbf{w}^\top \mathbf{x})). \tag{1}$$

If trained on data sampled from $P_S(X, Y)$, logistic regression models the distribution $P_S(Y|X)$ [13] through $P_h(Y = y | X = \mathbf{x}; \mathbf{w}) = (1 + e^{-\mathbf{w}^\top \mathbf{x} y})^{-1}$. In this paper, our goal is to adapt this classifier to the target distribution $P_T(Y|X)$.

## 3 Method

In this section, we begin with a semi-supervised approach and describe the rote-learning procedure to automatically annotate target domain inputs. The algorithm maintains and grows a training set that is iteratively adapted to the target domain. We then incorporate feature selection into the optimization, a crucial element of our domain-adaptation algorithm. The feature selection addresses the change in distribution and support from $P_S$ to $P_T$. Further, we introduce pseudo multi-view co-training [7, 10], which improves the rote-learning procedure by adding inputs with features that are still not used effectively by the current classifier. We use automated feature decomposition to artificially split our data into multiple views, explicitly to enable successful co-training.

### 3.1 Self-training for Domain Adaptation

First, we assume we are given a loss function $\ell$ – in our case the log-loss from eq. (1) – which provides some estimate of confidence in its predictions. In logistic regression, if $\hat{y} = sign(h(\mathbf{x}))$ is the prediction for an input $\mathbf{x}$, the probability $P_h(Y = \hat{y} | X = \mathbf{x}; \mathbf{w})$ is a natural metric of certainty (as $h(\mathbf{x})$ can be interpreted as a probability for $\mathbf{x}$ to be of label $+1$), but other methods [22] can be used. Self-training [19] is a simple and intuitive iterative algorithm to leverage unlabeled data. During training one maintains a labeled training set $L$ and an unlabeled test set $U$, initialized as $L = D_S \cup D_T^l$ and $U = D_T^u$. Each iteration, a classifier $h_\mathbf{w}$ is trained to minimize the loss function $\ell$ over $L$ and is evaluated on all elements of $U$. The $c$ most confident predictions on $U$ are *moved* to $L$ for the next iteration, labeled by the prediction of $sign(h_\mathbf{w})$. The algorithm terminates when $U$ is empty or all predictions are below a pre-defined confidence threshold (and considered unreliable). Algorithm 1 summarizes self-training in pseudo-code with the use of feature selection, described in the following section.

**Algorithm 1** SEDA pseudo-code.

---

1: Inputs: $L$ and $U$.
2: **repeat**
3:    $\mathbf{w}^* = \text{argmin}_{\mathbf{w}} \ell(\mathbf{w}; L) + \gamma s(L, U, \mathbf{w})$
4:    Apply $h_{\mathbf{w}^*}$ on all elements of $U$.
5:    Move up-to $c$ confident inputs $\mathbf{x}_i$ from $U$ to $L$, labeled as $sign(h(\mathbf{x}_i))$.
6: **until** No more predictions are confident
7: Return $h_{\mathbf{w}^*}$

---

## 3.2 Feature Selection

So far, we have not addressed that the two data sets $U$ and $L$ are *not* sampled from the same distribution. In domain adaptation, the training data is no longer representative of the test data. More explicitly, $P_S(Y|X = \mathbf{x})$ is different from $P_T(Y|X = \mathbf{x})$. For illustration, consider the sentiment analysis problem in section 4, where data consists of unigram and bigram bag-of-words features and the task is to classify if a book-review (source domain) or dvd-review (target domain) is positive or negative. Here, the bigram feature "must read" is indicative of a positive opinion within the source ("books") domain, but rarely appears in the target ("dvd") domain. A classifier, trained on the source-dominated set $L$, that relies too heavily on such features will not make enough high-confidence predictions on the set $U = D_T^u$.

To address this issue, we extend the classifier with a weighted $\ell_1$ regularization for feature selection. The weights are assigned to encourage the classifier to only use features that behave similarly in both $L$ and $U$. Different from previous work on feature selection for domain adaptation [25], where the goal is to find a new representation to minimize the difference between the distributions of the source and target domain, what we are proposing is to minimize the difference between the distributions of the labeled training set $L$ and the unlabeled set $U$ (which coincides with the testing set in our setting). This difference is crucial, as it makes the empirical distributions of $L$ and $U$ align *gradually*. For example, after some iterations, the classifier can pick features that are *never* present in the source domain, but which have entered $L$ through the rote-learning procedure.

We perform the feature selection implicitly through $\mathbf{w}$. For a feature $\alpha$, let us denote the Pearson correlation coefficient (PCC)[1] between feature value $\mathbf{x}_\alpha$ and the label $y$ for all pairs $(\mathbf{x}, y) \in L$ as $\rho_L(\mathbf{x}_\alpha, y)$. It can be shown that $\rho_L(\mathbf{x}_\alpha, y) \in [-1, 1]$ with a value of $+1$ if a feature is perfectly aligned with the label (i.e. the feature *is* the label), $0$ if it has no correlation, and $-1$ if it is of opposite polarity (*i.e.* the *inverted* label). Similarly, let us define the PCC for all pairs in $U$ as $\rho_{U;\mathbf{w}}(\mathbf{x}_\alpha, Y)$, where the unknown label $Y$ is a random variable drawn from the conditional probability $P_h(Y|X; \mathbf{w})$. The two PCC values indicate how predictive a feature is of the (estimated) class label in the two respective data sets. Ideally, we would like to choose features that are similarly predictive across the two sets. We measure how similarly a feature behaves across $L$ and $U$ with the product $\rho_L(\mathbf{x}_\alpha, y)\rho_{U;\mathbf{w}}(\mathbf{x}_\alpha, Y)$. With this notation, we define the feature weight that reflects the cross-domain *in*compatibility of a feature as

$$\Delta_{L,U,\mathbf{w}}(\alpha) = (1 - \rho_L(\mathbf{x}_\alpha, y)\rho_{U;\mathbf{w}}(\mathbf{x}_\alpha, Y)). \tag{2}$$

It is straight-forward to show that $\Delta_{L,U,\mathbf{w}} \in [0, 2]$. Intuitively, $\Delta_{L,U,\mathbf{w}}$ expresses to what degree we would like to *remove* a feature. A perfect feature, that is the label itself (and the prediction in $U$), results in a score of $0$. A feature that is not correlated with the class label in at least one of the two domains (and therefore is too domain-specific) obtains a score of $1$. A feature that switches polarization across domains (and therefore is "malicious") has a score $\Delta_{L,U,\mathbf{w}}(\alpha) > 1$ (in the extreme case if it is the label in $L$ and the *inverted* label in $U$, its score would be 2).

We incorporate (2) into a weighted $\ell_1$ regularization

$$s(L, U, \mathbf{w}) = \sum_{\alpha=1}^{d} \Delta_{L,U,\mathbf{w}}(\alpha)|\mathbf{w}_\alpha|. \tag{3}$$

Intuitively (3) encourages feature sparsity with a strong emphasis on features with little or opposite correlation across the domains, whereas good features that are consistently predictive in both

domains become *cheap*. We refer to this version of the algorithm as Self-training for Domain Adaptation (SEDA). The optimization with feature selection, used in Algorithm 1, becomes

$$\mathbf{w} = \text{argmin}_{\mathbf{w}} \ell(L) + \gamma s(L, U, \mathbf{w}). \tag{4}$$

Here, $\gamma \geq 0$ denotes the loss-regularization trade-off parameter. As we have very few labeled inputs from the *target* domain in the early iterations, stronger regularization is imposed so that only features shared across the two domains are used. When more and more inputs from the *target* domain are included in the training set, we gradually decrease the regularization to accommodate *target* specific features. The algorithm is very insensitive to the exact initial choice of $\gamma$. The guideline is to start with a relatively large number, and decrease it until the selected feature set is not empty. In our implementation, we set it to $\gamma_0 = 0.1$, and we divide it by a factor of $1.1$ during each iteration.

### 3.3    Co-training for Domain Adaptation

For rote-learning to be effective, we need to move test inputs from $U$ to $L$ that 1) are correctly classified (with high probability) and 2) have potential to improve the classifier in future iterations. The former is addressed by the feature selecting regularization from the previous section – restricting the classifier to a sub-set of features that are known to be cross-data set compatible reduces the generalization error on $U$. In this section we address the second requirement. We want to add inputs $\mathbf{x}_i$ that contain additional features, which were not used to obtain the prediction $h_{\mathbf{w}}(\mathbf{x}_i)$ and would enrich the training set $L$.

If the exact labels of the inputs in $U$ were known, a good active learning [26] strategy would be to move inputs to $L$ on which the current classifier $h_{\mathbf{w}}$ is *most uncertain*. In our setting, this would be clearly ill advised as the uncertain prediction is also used as the label. A natural solution to this dilemma is co-training [7]. Co-training assumes the data set is presented in two separate views and two classifiers are trained, one in each view. Each iteration, only inputs that are confident according to *exactly one* of the two classifiers are moved to the training set. This way, one classifier provides the (estimated) labels to the inputs on which the *other* classifier is uncertain.

In our setting we *do not* have multiple views and which features are selected varies in each iteration. Hence, co-training does not apply out-of-the-box. We can, however, *split* our features into two mutually exclusive views such that co-training is effective. To this end we follow the pseudo-multiview regularization introduced by Chen *et al.* [10]. The main intuition is to train *two* classifiers on a single view $\mathcal{X}$ such that: (1) both perform well on the labeled data; (2) both are trained on strictly different features; (3) together they are likely to satisfy Balcan's condition of $\epsilon$-expandability [2], a necessary and sufficient pre-condition for co-training to work[2]. These three aspects can be formulated explicitly as three modifications of our optimization problem (4). We discuss each of them in detail in the following.

**Loss.** Two classifiers are required for co-training, whose weight vectors we denote by $\mathbf{u}$ and $\mathbf{v}$. The performance of each classifier is measured by the log-loss $\ell(\cdot; L)$ in eq. (1). To ensure that both classifiers perform well on the training set $L$, *i.e.* both have a small training loss, we train them jointly while minimizing the soft-maximum[3] of the two losses,

$$\log \left( e^{\ell(\mathbf{u}; L)} + e^{\ell(\mathbf{v}; L)} \right). \tag{5}$$

**Feature Decomposition.** Co-training requires the two classifiers to be trained on different feature spaces. We create those by splitting the feature-space into two mutually exclusive sub-sets. More precisely, for each feature $\alpha$, at least one of the two classifiers must have a zero weight in the $\alpha^{th}$ dimension. We can enforce this across all features with the equality constraint

$$\sum_{\alpha=1}^{d} \mathbf{u}_{\alpha}^2 \mathbf{v}_{\alpha}^2 = 0. \tag{6}$$

$\epsilon$-**Expandability.** In the original co-training formulation [7], it is assumed that the two views of the data are class conditionally independent. This assumption is very strong and can easily be

violated in practice [20]. Recent work [2] weakens this requirement significantly to a condition of $\epsilon$-*expandability*. Loosely phrased, for the two classifiers to be able to teach each other, they must make confident predictions on different subsets of the unlabeled set $U$.

For the classifier $h_{\mathbf{u}}$, let $\hat{y} = \text{sign}(\mathbf{u}^\top \mathbf{x}) \in \{\pm 1\}$ denote the class prediction and $P_h(\hat{y}|\mathbf{x}; \mathbf{u})$ its confidence. Define $c_{\mathbf{u}}(\mathbf{x})$ as a confidence indicator function (for some confidence threshold $\tau > 0$)[4]

$$c_{\mathbf{u}}(\mathbf{x}) = \left\{ \begin{array}{ll} 1 & \text{if } p(\hat{y}|\mathbf{x}; \mathbf{u}) > \tau \\ 0 & \text{otherwise,} \end{array} \right. \tag{7}$$

and $c_{\mathbf{v}}$ respectively. Then the $\epsilon$-expanding condition translates to

$$\sum_{\mathbf{x} \in U} [c_{\mathbf{u}}(\mathbf{x}) \bar{c}_{\mathbf{v}}(\mathbf{x}) + \bar{c}_{\mathbf{u}}(\mathbf{x}) c_{\mathbf{v}}(\mathbf{x})] \geq \epsilon \min \left[ \sum_{\mathbf{x} \in U} c_{\mathbf{u}}(\mathbf{x}) c_{\mathbf{v}}(\mathbf{x}), \sum_{\mathbf{x} \in U} \bar{c}_{\mathbf{u}}(\mathbf{x}) \bar{c}_{\mathbf{v}}(\mathbf{x}) \right], \tag{8}$$

for some $\epsilon > 0$. Here, $\bar{c}_{\mathbf{u}}(\mathbf{x}) = 1 - c_{\mathbf{u}}(\mathbf{x})$ indicates that classifier $h_{\mathbf{u}}$ is *not* confident about input $\mathbf{x}$. Intuitively, the constraint in eq. (8) ensures that the total number of inputs in $U$ that can be used for rote-learning because *exactly one* classifier is confident (LHS), is larger than the set of inputs which *cannot* be used because *both* classifiers are already confident or *both are not* confident (RHS).

In summary, the framework splits the feature space into two mutually exclusive sub-sets. This representation enables us to train two logistic regression classifiers, both with small loss on the labeled data set, while satisfying two constraints to ensure feature decomposition and $\epsilon$-expandability. Our final classifier has the weight vector $\mathbf{w} = \mathbf{u} + \mathbf{v}$. We refer to the resulting algorithm as CODA (Co-training for Domain Adaptation), which can be stated concisely with the following optimization problem:

$$
\begin{array}{l}
\min_{\mathbf{w}, \mathbf{u}, \mathbf{v}} \quad \log \left( e^{\ell(\mathbf{u}; L)} + e^{\ell(\mathbf{v}; L)} \right) + \gamma s(L, U, \mathbf{w}) \\[4pt]
\textbf{subject to:} \\
\textbf{(1)} \ \sum_{i=1}^{d} \mathbf{u}_i^2 \mathbf{v}_i^2 = 0 \\
\textbf{(2)} \ \sum_{\mathbf{x} \in U} [c_{\mathbf{u}}(\mathbf{x}) \bar{c}_{\mathbf{v}}(\mathbf{x}) + \bar{c}_{\mathbf{u}}(\mathbf{x}) c_{\mathbf{v}}(\mathbf{x})] \geq \epsilon \min \left[ \sum_{\mathbf{x} \in U} c_{\mathbf{u}}(\mathbf{x}) c_{\mathbf{v}}(\mathbf{x}), \sum_{\mathbf{x} \in U} \bar{c}_{\mathbf{u}}(\mathbf{x}) \bar{c}_{\mathbf{v}}(\mathbf{x}) \right] \\
\textbf{(3)} \ \mathbf{w} = \mathbf{u} + \mathbf{v}
\end{array}
$$

The optimization is non-convex. However, as it is not particularly sensitive to initialization, we set $\mathbf{u}, \mathbf{v}$ randomly and optimize with standard conjugate gradient descent[5]. Due to space constraints we do not include a pseudo-code implementation of CODA. The implementation is essentially identical to that of SEDA (Algorithm 1) where the above optimization problem is solved instead of eq. (4) in line 3. In line 5, we move inputs that one classifier is confident about while the other one is uncertain to the training set $L$ to improve the classifier in future iterations.

## 4 Results

We evaluate our algorithm together with several other domain adaptation algorithms on the "Amazon reviews" benchmark data sets [6]. The data set contains reviews of four different types of products: books, DVDs, electronics, and kitchen appliances from Amazon.com. In the original dataset, each review is associated with a rating of 1-5 stars. For simplicity, we are only concerned about whether or not a review is positive (higher than 3 stars) or negative (3 stars or lower). That is, $y_i = \{+1, -1\}$, where $y_i = 1$ indicates that it is a positive review, and $-1$ otherwise. The data from four domains results in 12 directed adaptation tasks (*e.g. books $\rightarrow$ dvds*). Each domain adaptation task consists of $2,000$ labeled source inputs and around $4,000$ unlabeled target test inputs (varying slightly between tasks). We let the amount of labeled target data vary from $0$ to $1600$. For each setting with target labels we ran 10 experiments with different, randomly chosen, labeled instances. The original feature space of unigrams and bigrams is on average approximately $100,000$ dimensions across

different domains. To reduce the dimensionality, we only use features that appear at least 10 times in a particular domain adaptation task (with approximately $40,000$ features remaining). Further, we pre-process the data set with standard tf-idf [24] feature re-weighting.

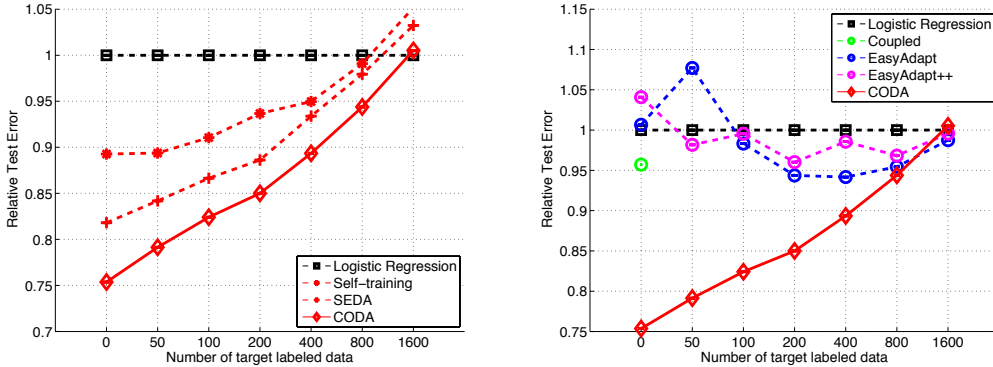

Figure 1: Relative test-error reduction over logistic regression, averaged across all 12 domain adaptation tasks, as a function of the target training set size. *Left:* A comparison of the three algorithms from section 3. The graph shows clearly that self-training (Self-training vs. Logistic Regression), feature-selection (SEDA vs. Self-training) and co-training (CODA vs. SEDA), each improve the accuracy substantially. *Right:* A comparison of CODA with four state-of-the-art domain adaptation algorithms. CODA leads to particularly strong improvements under little target supervision.

As a first experiment, we compare the three algorithms from Section 3 and logistic regression as a baseline. The results are in the left plot of figure 1. For **logistic regression**, we ignore the difference between source and target distribution, and train a classifier on the union of both labeled data sets. We use $\ell_2$ regularization, and set the regularization constant with 5-fold cross-validation. In figure 1, all classification errors are shown relative to this baseline. Our second baseline is **self-training**, which adds self-training to logistic regression – as described in section 3.1. We start with the set of labeled instances from source and target domain, and gradually add confident predictions to the training set from the unlabeled target domain (without regularization). **SEDA** adds feature selection to the self-training procedure, as described in section 3.2. We optimize over 100 iterations of self-training, at which stage the regularization was effectively zero and the classifier converged. For **CODA** we replace self-training with pseudo-multi-view co-training, as described in section 3.3.

The left plot in figure 1 shows the relative classification errors of these four algorithms averaged over all 12 domain adaptation tasks, under varying amounts of target labels. We observe two trends: First, there are clear gaps between logistic regression, self-training, SEDA, and CODA. From these three gaps one can conclude that self-training, feature-selection and co-training each lead to substantial improvements in classification error. A second trend is that the relative improvement over logistic regression reduces as more labeled target data becomes available. This is not surprising, as with sufficient target labels the task turns into a classical supervised learning problem and the source data becomes irrelevant.

As a second experiment, we compare CODA against three state-of-the-art domain adaptation algorithms. We refer to these as **Coupled**, the coupled-subspaces approach [6], **EasyAdapt** [11], and **EasyAdapt++.** [15]. Details about the respective algorithms are provided in section 5. Coupled subspaces, as described in [6], does not utilize labeled target data and its result is depicted as a single point. The right plot in figure 1 compares these algorithms, relative to logistic regression. Figure 3 shows the individual results on all the 12 adaptation tasks with absolute classification error rates. The error bars show the standard deviation across the 10 runs with different labeled instances. EasyAdapt and EasyAdapt++, both consistently improve over logistic regression once sufficient target data is available. It is noteworthy that, on average, CODA outperforms the other algorithms in almost all settings when 800 labeled target points or less are present. With 1600 labeled target points all algorithms perform similar to the baseline and additional source data is irrelevant. All hyper-parameters of competing algorithms were carefully set by 5-fold cross validation.

Concerning computational requirements, it is fair to say that CODA is significantly slower than the other algorithms, as each iteration is of comparable complexity as logistic regression or EasyAdapt.

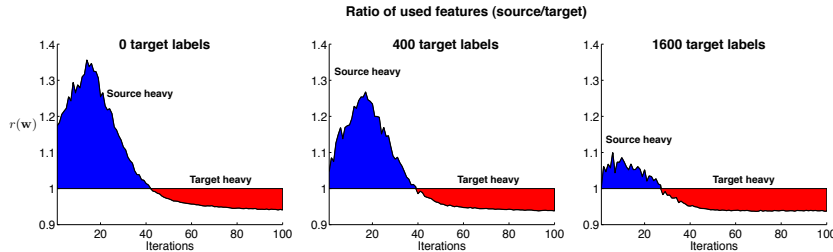

Figure 2: The ratio of the average number of used features between source and target inputs (9), tracked throughout the CODA optimization. The three plots show the same statistic at different amounts of target labels. Initially, an input from the source domain has on average 10-35% more features that are used by the classifier than a target input. At around iteration 40, this relation changes and the classifier uses more target-typical features. The graph shows the geometric mean across all adaptation tasks. With no target data available (left plot), the early spike in source dominance is more pronounced and decreases when more target labels are available (middle and right plot).

In typical domain adaptation settings this is generally not a problem, as training sets tend to be small. In our experiments, the average training time for CODA[6] was about 20 minutes.

Finally, we investigate the feature-selection process during CODA training. Let us define the indicator function $\delta(a) \in \{0, 1\}$ to be $\delta(a) = 0$ if and only if $a = 0$, which operates element-wise on vectors. The vector $\delta(\mathbf{w}) \in \{0, 1\}^d$ indicates which features are used in the classifier and $\delta(\mathbf{x}_i)$ indicates which features are present in input $\mathbf{x}_i$. We can denote the ratio between the average number of *used* features in labeled training inputs over those in unlabeled target inputs as

$$r(\mathbf{w}) = \frac{\frac{1}{|D_S^l|} \sum_{\mathbf{x}_s \in D_S^l} \delta(\mathbf{w})^\top \delta(\mathbf{x}_s)}{\frac{1}{|D_T^l|} \sum_{\mathbf{x}_t \in D_T^l} \delta(\mathbf{w})^\top \delta(\mathbf{x}_t)}. \tag{9}$$

Figure 2 shows the plot of $r(\mathbf{w})$ for all weight vectors during the 100 iterations of CODA, averaged across all 12 data sets. The three plots show the same statistic under varying amounts of target labels. Two trends can be observed: First, during CODA training, the classifier initially selects more source-specific features. For example in the case with zero labeled target data, during early iterations the average source input contains $20 - 35\%$ more *used* features relative to target inputs. This source-heavy feature distribution changes and eventually turns into target-heavy distribution as the classifier adapts to the target domain. As a second trend, we observe that with more target labels (right plot), this spike in source features is much less pronounced whereas the final target-heavy ratio is unchanged but starts earlier. This indicates that as the target labels increase, the classifier makes less use of the source data and relies sooner and more directly on the target signal.

## 5 Related Work and Discussion

Domain adaptation algorithms that do not use labeled target domain data are sometimes called *unsupervised* adaptation algorithms. There are roughly three types of algorithms in this group. The first type, which includes the coupled subspaces algorithm of Blitzer *et al.* [5], learns a shared representation under which the source and target distributions are closer than under the ambient feature space [28]. The largest disadvantage of these algorithms is that they do not jointly optimize the predictor and the representation, which prevents them from focusing on those features which are both different *and* predictive. By jointly optimizing the feature selection, the multi-view split and the prediction, CODA allows us to do both.

The second type of algorithm attempts to directly minimize the divergence between domains, typically by weighting individual instances [14, 16, 18]. These algorithms do not assume highly divergent domains (e.g. those with unique target features), but they have the advantage over both CODA and representation-learning of learning asymptotically optimal target predictors from only

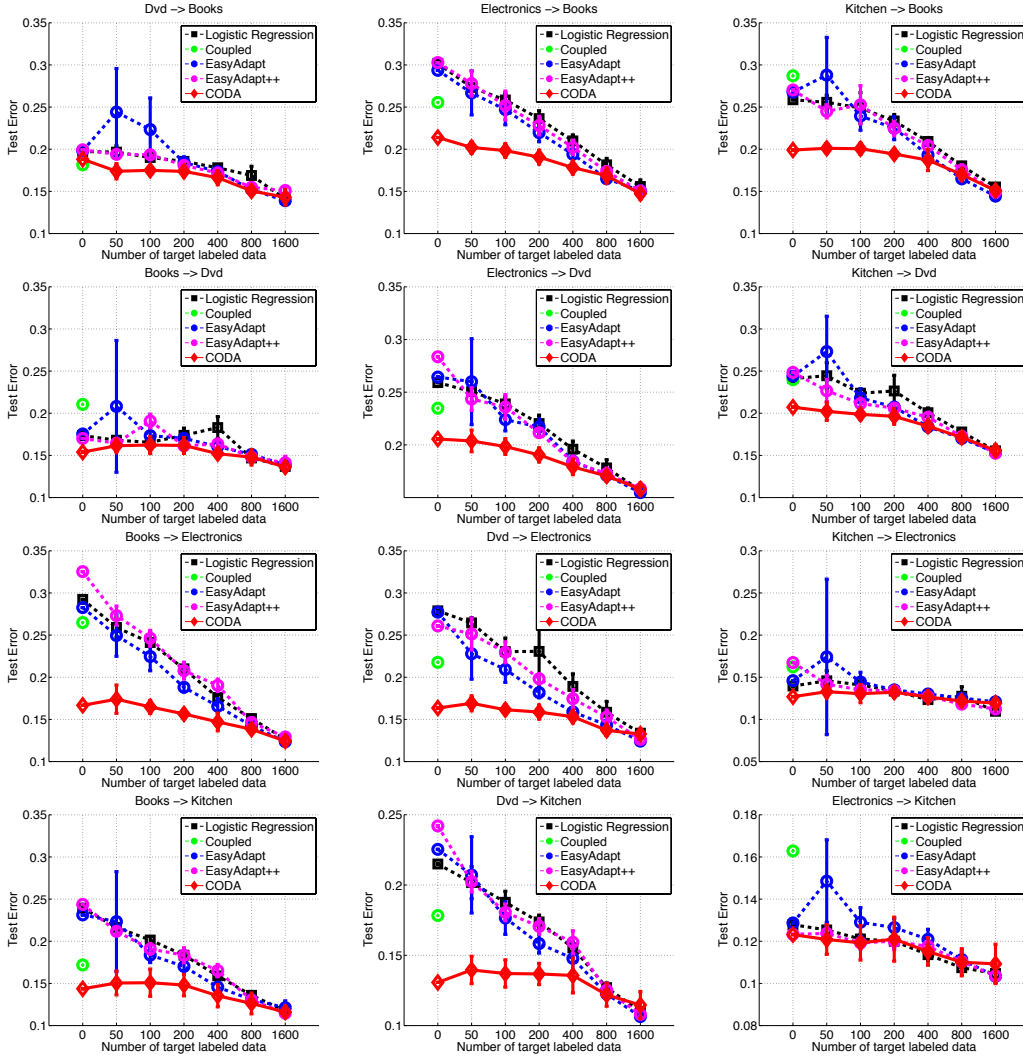

Figure 3: The individual results on all domain adaptation tasks under varying amounts of labeled target data. The graphs show the absolute classification error rates. All settings with existing labeled target data were averaged over 10 runs (with randomly selected labeled instances). The vertical bars indicate the standard deviation in these cases.

source training data (when their assumptions hold). We did not explore them here because their assumptions are clearly violated for this data set.

In natural language processing, a final type of very successful algorithm self-trains on its own target predictions to automatically annotate new target domain features [19]. These methods are most closely related, in spirit, to our own CODA algorithm. Indeed, our self-training baseline is intended to mimic this style of algorithm.

The final set of domain adaptation algorithms, which we compared against but did not describe, are those which actively seek to minimize the labeling divergence between domains using multi-task techniques [1, 8, 9, 12, 21, 27]. Most prominently, Daumé [11] trains separate source and target models, but regularizes these models to be close to one another. The EasyAdapt++ variant of this algorithm, which we compared against, generalizes this to the semi-supervised setting by making the assumption that for unlabeled target instances, the tasks should be similar. Although these methods did not significantly out-perform our baselines in the sentiment data set, we note that there do exist data sets on which such multi-task techniques are especially important [11], and we hope soon to explore combinations of CODA with multi-task learning on those data sets.

## Footnotes

[1]The PCC for two random variables $X, Y$ is defined as $\rho = \frac{E[(X-\mu_X)(Y-\mu_Y)]}{\sigma_X \sigma_Y}$, where $\mu_X$ denotes the mean and $\sigma_X$ the standard deviation of $X$.

[2]Provided that the classifiers are never confident and wrong — which can be violated in practice.

[3]The soft-max of a set of elements $S$ is a differentiable approximation of $\max(S) \approx \log(\sum_{s \in S} e^s)$.

[4]In our implementation, the 0-1 indicator was replaced by a very steep differentiable sigmoid function, and $\tau$ was set to 0.8 across different experiments.

[5]We use minimize.m (http://tinyurl.com/minimize-m).

[6]We used a straight-forward Matlab$^{TM}$ implementation.

# References

[1] R.K. Ando and T. Zhang. A framework for learning predictive structures from multiple tasks and unlabeled data. *The Journal of Machine Learning Research*, 6:1817–1853, 2005.

[2] M.F. Balcan, A. Blum, and K. Yang. Co-training and expansion: Towards bridging theory and practice. *NIPS*, 17:89–96, 2004.

[3] S. Ben-David, J. Blitzer, K. Crammer, A. Kulesza, F. Pereira, and Jenn Wortman. A theory of learning from different domains. *Machine Learning*, 2009.

[4] J. Blitzer, M. Dredze, and F. Pereira. Biographies, bollywood, boom-boxes and blenders: Domain adaptation for sentiment classification. In *Association for Computational Linguistics*, Prague, Czech Republic, 2007.

[5] J. Blitzer, D. Foster, and S. Kakade. Domain adaptation with coupled subspaces. In *Conference on Artificial Intelligence and Statistics*, Fort Lauterdale, 2011.

[6] J. Blitzer, R. McDonald, and F. Pereira. Domain adaptation with structural correspondence learning. In *Proceedings of the 2006 Conference on Empirical Methods in Natural Language Processing*, pages 120–128. Association for Computational Linguistics, 2006.

[7] A. Blum and T. Mitchell. Combining labeled and unlabeled data with co-training. In *Proceedings of the eleventh annual conference on Computational learning theory*, page 100. ACM, 1998.

[8] R. Caruana. Multitask learning. *Machine Learning*, 28:41–75, 1997.

[9] O. Chapelle, P. Shivaswamy, S. Vadrevu, K.Q. Weinberger, Y. Zhang, and B. Tseng. Multi-task learning for boosting with application to web search ranking. In *Proceedings of the 16th ACM SIGKDD international conference on Knowledge discovery and data mining*, KDD '10, pages 1189–1198, New York, NY, USA, 2010. ACM.

[10] M. Chen, K.Q. Weinberger, and Y. Chen. Automatic Feature Decomposition for Single View Co-training. In *International Conference on Machine Learning*, 2011.

[11] H. Daume III. Frustratingly easy domain adaptation. In *Association for Computational Linguistics*, 2007.

[12] T. Evgeniou, C.A. Micchelli, and M. Pontil. Learning multiple tasks with kernel methods. *Journal of Machine Learning Research*, 6(1):615, 2006.

[13] T. Hastie, R. Tibshirani, and J. Friedman. *The Elements of Statistical Learning*. Springer Verlag, New York, 2009.

[14] J. Huang, A.J. Smola, A. Gretton, K. M. Borgwardt, and B. Scholkopf. Correcting sample selection bias by unlabeled data. In *NIPS 19*, pages 601–608. MIT Press, Cambridge, MA, 2007.

[15] H. Daume III, A. Kumar, and A. Saha. Co-regularization based semi-supervised domain adaptation. In *NIPS 23*, pages 478–486. MIT Press, 2010.

[16] J. Jiang and C.X. Zhai. Instance weighting for domain adaptation in nlp. In *Proceedings of the 45th Annual Meeting of the Association of Computational Linguistics*, pages 264–271, Prague, Czech Republic, June 2007. Association for Computational Linguistics.

[17] Qian Liu, Aaron Mackey, David Roos, and Fernando Pereira. Evigan: a hidden variable model for integrating gene evidence for eukaryotic gene prediction. *Bioinformatics*, 2008.

[18] T. Mansour, M. Mohri, and A. Rostamizadeh. Domain adaptation with multiple sources. In *NIPS 21*, pages 1041–1048. MIT Press, 2009.

[19] D. McClosky, E. Charniak, and M. Johnson. Reranking and self-training for parser adaptation. In *Proceedings of the 21st International Conference on Computational Linguistics and the 44th annual meeting of the Association for Computational Linguistics*, pages 337–344. Association for Computational Linguistics, 2006.

[20] K. Nigam and R. Ghani. Analyzing the effectiveness and applicability of co-training. In *Proceedings of the ninth international conference on Information and knowledge management*, pages 86–93. ACM, 2000.

[21] S. Parameswaran and K.Q. Weinberger. Large margin multi-task metric learning. In *NIPS 23*, pages 1867–1875. 2010.

[22] J.C. Platt et al. Probabilities for sv machines. *NIPS*, pages 61–74, 1999.

[23] K. Saenko, B. Kulis, M. Fritz, and T. Darrell. Adapting visual category models to new domains. *Computer Vision–ECCV 2010*, pages 213–226, 2010.

[24] G. Salton and C. Buckley. Term-weighting approaches in automatic text retrieval. *Information processing & management*, 24(5):513–523, 1988.

[25] S. Satpal and S. Sarawagi. Domain adaptation of conditional probability models via feature subsetting. *Knowledge Discovery in Databases: PKDD 2007*, pages 224–235, 2007.

[26] B. Settles. Active learning literature survey. *Machine Learning*, 15(2):201–221, 1994.

[27] K.Q. Weinberger, A. Dasgupta, J. Langford, A. Smola, and J. Attenberg. Feature hashing for large scale multitask learning. In *Proceedings of the 26th Annual International Conference on Machine Learning*, pages 1113–1120. ACM, 2009.

[28] G. Xue, W. Dai, Q. Yang, and Y. Yu. Topic-bridged plsa for cross-domain text classication. In *SIGIR*, 2008.

